# Putting Bayes to sleep

**Wouter M. Koolen**[*]    **Dmitry Adamskiy**[†]    **Manfred K. Warmuth**[‡]

## Abstract

We consider sequential prediction algorithms that are given the predictions from a set of models as inputs. If the nature of the data is changing over time in that different models predict well on different segments of the data, then adaptivity is typically achieved by mixing into the weights in each round a bit of the initial prior (kind of like a weak restart). However, what if the favored models in each segment are from a *small subset*, i.e. the data is likely to be predicted well by models that predicted well before? Curiously, fitting such "sparse composite models" is achieved by mixing in a bit of all the past posteriors. This self-referential updating method is rather peculiar, but it is efficient and gives superior performance on many natural data sets. Also it is important because it introduces a long-term memory: any model that has done well in the past can be recovered quickly. While Bayesian interpretations can be found for mixing in a bit of the initial prior, no Bayesian interpretation is known for mixing in past posteriors.

We build atop the "specialist" framework from the online learning literature to give the Mixing Past Posteriors update a proper Bayesian foundation. We apply our method to a well-studied multitask learning problem and obtain a new intriguing efficient update that achieves a significantly better bound.

## 1 Introduction

We consider sequential prediction of outcomes $y_1, y_2, \ldots$ using a set of models $m = 1, \ldots, M$ for this task. In practice $m$ could range over a mix of human experts, parametric models, or even complex machine learning algorithms. In any case we denote the prediction of model $m$ for outcome $y_t$ given past observations $y_{<t} = (y_1, \ldots, y_{t-1})$ by $P(y_t|y_{<t}, m)$. The goal is to design a computationally efficient predictor $P(y_t|y_{<t})$ that maximally leverages the predictive power of these models as measured in log loss. The yardstick in this paper is a notion of *regret* defined w.r.t. a given *comparator class* of models or composite models: it is the additional loss of the predictor over the best comparator. For example if the comparator class is the set of base models $m = 1, \ldots, M$, then the regret for a sequence of $T$ outcomes $y_{\leq T} = (y_1, \ldots, y_T)$ is

$$\mathcal{R} := \sum_{t=1}^{T} -\ln P(y_t|y_{<t}) \;-\; \min_{m=1}^{M} \sum_{t=1}^{T} -\ln P(y_t|y_{<t}, m).$$

The Bayesian predictor (detailed below) with uniform model prior has regret at most $\ln M$ for all $T$.

Typically the nature of the data is changing with time: in an initial segment one model predicts well, followed by a second segment in which another model has small loss and so forth. For this scenario the natural comparator class is the set of *partition models* which divide the sequence of $T$ outcomes into $B$ segments and specify the model that predicts in each segment. By running Bayes on all exponentially many partition models comprising the comparator class, we can guarantee regret $\ln \binom{T-1}{B-1} + B \ln M$, which is optimal. The goal then is to find *efficient* algorithms with approximately

---

[*]Supported by NWO Rubicon grant 680-50-1010.

[†]Supported by Veterinary Laboratories Agency of Department for Environment, Food and Rural Affairs.

[‡]Supported by NSF grant IIS-0917397.

the same guarantee as full Bayes. In this case this is achieved by the Fixed Share [HW98] predictor. It assigns a certain prior to all partition models for which the exponentially many posterior weights collapse to $M$ posterior weights that can be maintained efficiently. Modifications of this algorithm achieve essentially the same bound for all $T$, $B$ and $M$ simultaneously [VW98, KdR08].

In an open problem Yoav Freund [BW02] asked whether there are algorithms that have small regret against *sparse* partition models where the base models allocated to the segments are from a small subset of $N$ of the $M$ models. The Bayes algorithm when run on all such partition models achieves regret $\ln \binom{M}{N} + \ln \binom{T-1}{B-1} + B \ln N$, but contrary to the non-sparse case, emulating this algorithm is NP-hard. However in a breakthrough paper, Bousquet and Warmuth in 2001 [BW02] gave the efficient MPP algorithm with only a slightly weaker regret bound. Like Fixed Share, MPP maintains $M$ "posterior" weights, but it instead mixes in a bit of all past posteriors in each update. This causes weights of previously good models to "glow" a little bit, even if they perform bad locally. When the data later favors one of those good models, its weight is pulled up quickly. However the term "posterior" is a misnomer because no Bayesian interpretation for this curious self-referential update was known. Understanding the MPP update is a very important problem because in many practical applications [HLSS00, GWBA02][1] it significantly outperforms Fixed Share.

Our main philosophical contribution is finding a Bayesian interpretation for MPP. We employ the specialist framework from online learning [FSSW97, CV09, CKZV10]. So-called *specialist* models are either *awake* or *asleep*. When they are awake, they predict as usual. However when they are asleep, they "go with the rest", i.e. they predict with the combined prediction of all awake models.

Instead of fully coordinated partition models, we construct *partition specialists* consisting of a base model and a set of segments where this base model is awake. The figure to the right shows how a comparator partition model is assembled from partition specialists. We can emulate Bayes on all partition specialists; NP-completeness is avoided by forgoing a-priori segment synchronization. By carefully choosing the prior, the exponentially many posterior weights collapse to the small number of weights used by the efficient MPP algorithm. Our analysis technique magically aggregates the contribution of the $N$ partition specialists that constitute the comparator partition, showing that we achieve regret close to the regret of Bayes when run on all full partition models. Actually our new insights into the nature of MPP result in slightly improved regret bounds.

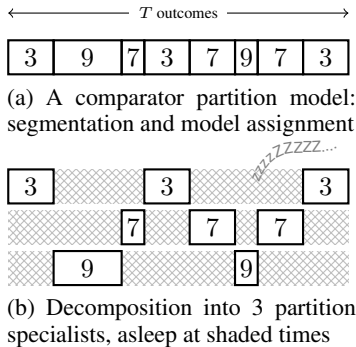

(a) A comparator partition model: segmentation and model assignment

(b) Decomposition into 3 partition specialists, asleep at shaded times

We then apply our methods to an online multitask learning problem where a small subset of models from a big set solve a large number of tasks. Again simulating Bayes on all sparse assignments of models to tasks is NP-hard. We split an assignment into *subset specialists* that assign a single base model to a subset of tasks. With the right prior, Bayes on these subset specialists again gently collapses to an efficient algorithm with a regret bound not much larger than Bayes on all assignments. This considerably improves the previous regret bound of [ABR07]. Our algorithm simply maintains one weight per model/task pair and does not rely on sampling (often used for multitask learning).

Why is this line of research important? We found a new intuitive Bayesian method to quickly recover information that was learned before, allowing us to exploit sparse composite models. Moreover, it expressly avoids computational hardness by splitting composite models into smaller constituent "specialists" that are asleep in time steps outside their jurisdiction. This method clearly beats Fixed Share when *few* base models constitute a partition, i.e. the composite models are sparse.

We expect this methodology to become a main tool for making Bayesian prediction adapt to sparse models. The goal is to develop general tools for adding this type of adaptivity to existing Bayesian models without losing efficiency. It also lets us look again at the updates used in Nature in a new light, where species/genes cannot dare adapt too quickly to the current environment and must guard themselves against an environment that changes or fluctuates at a large scale. Surprisingly these type of updates might now be amenable to a Bayesian analysis. For example, it might be possible to interpret sex and the double stranded recessive/dominant gene device employed by Nature as a Bayesian update of genes that are either awake or asleep.

## 2 Bayes and Specialists

We consider sequential prediction of outcomes $y_1, y_2, \ldots$ from a finite alphabet. Assume that we have access to a collection of models $m = 1, \ldots, M$ with data likelihoods $P(y_1, y_2, \ldots | m)$. We then design a prior $P(m)$ with roughly two goals in mind: the Bayes algorithm should "collapse" (become efficient) and have a good regret bound. After observing past outcomes $y_{<t} := (y_1, \ldots, y_{t-1})$, the next outcome $y_t$ is predicted by the *predictive distribution* $P(y_t | y_{<t})$, which averages the model predictions $P(y_t | y_{<t}, m)$ according to the *posterior distribution* $P(m | y_{<t})$:

$$P(y_t | y_{<t}) = \sum_{m=1}^{M} P(y_t | y_{<t}, m) P(m | y_{<t}), \quad \text{where} \quad P(m | y_{<t}) = \frac{P(y_{<t} | m) P(m)}{P(y_{<t})}.$$

The latter is conveniently updated step-wise: $P(m | y_t, y_{<t}) = P(y_t | y_{<t}, m) P(m | y_{<t}) / P(y_t | y_{<t})$.

The log loss of the Bayesian predictor on data $y_{\leq T} := (y_1, \ldots, y_T)$ is the cumulative loss of the predictive distributions and this readily relates to the cumulative loss of any model $\hat{m}$:

$$\underbrace{-\ln P(y_{\leq T})}_{\sum_{t=1}^{T} -\ln P(y_t | y_{<t})} - \underbrace{\left(-\ln P(y_{\leq T} | \hat{m})\right)}_{\sum_{t=1}^{T} -\ln P(y_t | y_{<t}, \hat{m})} = -\ln\Big(\sum_{m=1}^{M} P(y_{\leq T} | m) P(m)\Big) - \left(-\ln P(y_{\leq T} | \hat{m})\right) \leq -\ln P(\hat{m}).$$

That is, the additional loss (or *regret*) of Bayes w.r.t. model $\hat{m}$ is at most $-\ln P(\hat{m})$. The uniform prior $P(m) = 1/M$ ensures regret at most $\ln M$ w.r.t. any model $\hat{m}$. This is a so-called *individual sequence* result, because no probabilistic assumptions were made on the data.

Our main results will make essential use of the following fancier weighted notion of regret. Here $U(m)$ is any distribution on the models and $\triangle\big(U(m)\|P(m)\big)$ denotes the relative entropy $\sum_{m=1}^{M} U(m) \ln \frac{U(m)}{P(m)}$ between the distributions $U(m)$ and $P(m)$:

$$\sum_{m=1}^{M} U(m)\big(-\ln P(y_{\leq T}) - (-\ln P(y_{\leq T} | m))\big) = \triangle\big(U(m)\|P(m)\big) - \triangle\big(U(m)\|P(m | y_{\leq T})\big). \quad (1)$$

By dropping the subtracted positive term we get an upper bound. The previous regret bound is now the special case when $U$ is concentrated on model $\hat{m}$. However when multiple models are good we achieve tighter regret bounds by letting $U$ be the uniform distribution on all of them.

**Specialists**   We now consider a complication of the prediction task, which was introduced in the online learning literature under the name *specialists* [FSSW97]. The Bayesian algorithm, adapted to this task, will serve as the foundation of our main results. The idea is that in practice the predictions $P(y_t | y_{<t}, m)$ of some models may be unavailable. Human forecasters may be specialized, unreachable or too expensive, algorithms may run out of memory or simply take too long. We call models that may possibly abstain from prediction *specialists*. The question is how to produce quality predictions from the predictions that are available.

We will denote by $W_t$ the set of specialists whose predictions are available at time $t$, and call them *awake* and the others *asleep*. The crucial idea, introduced in [CV09], is to assign to the sleeping specialists the prediction $P(y_t | y_{<t})$. But wait! That prediction $P(y_t | y_{<t})$ is defined to average all model predictions, including those of the sleeping specialists, which we just defined to be $P(y_t | y_{<t})$:

$$P(y_t | y_{<t}) = \sum_{m \in W_t} P(y_t | y_{<t}, m) \, P(m | y_{<t}) + \sum_{m \notin W_t} P(y_t | y_{<t}) \, P(m | y_{<t}).$$

Although this equation is self-referential, it does have a unique solution, namely

$$P(y_t | y_{<t}) := \frac{\sum_{m \in W_t} P(y_t | y_{<t}, m) P(m | y_{<t})}{P(W_t | y_{<t})}.$$

Thus the sleeping specialists are assigned the average prediction of the awake ones. This completes them to full models to which we can apply the unaltered Bayesian method as before. At first this may seem like a kludge, but actually this phenomenon arises naturally wherever concentrations are

manipulated. For example, in a democracy abstaining essentially endorses the vote of the participating voters or in Nature unexpressed genes reproduce at rates determined by the active genes of the organism. The effect of abstaining on the update of the posterior weights is also intuitive: weights of asleep specialists are unaffected, whereas weights of awake models are updated with Bayes rule and then renormalised to the original weight of the awake set:

$$P(m|y_{\leq t}) = \begin{cases} \frac{P(y_t|y_{<t},m)P(m|y_{<t})}{P(y_t|y_{<t})} = \frac{P(y_t|y_{<t},m)P(m|y_{<t})}{\sum_{m \in W_t} P(y_t|y_{<t},m)P(m|y_{<t})} \, P(W_t|y_{<t}) & \text{if } m \in W_t, \\ \frac{\cancel{P(y_t|y_{<t})}P(m|y_{<t})}{\cancel{P(y_t|y_{<t})}} = P(m|y_{<t}) & \text{if } m \notin W_t. \end{cases} \quad (2)$$

To obtain regret bounds in the specialist setting, we use the fact that sleeping specialists $m \notin W_t$ are defined to predict $P(y_t|y_{<t}, m) := P(y_t|y_{<t})$ like the Bayesian aggregate. Now (1) becomes:

**Theorem 1** ([FSSW97, Theorem 1]). *Let $U(m)$ be any distribution on a set of specialists with wake sets $W_1, W_2, \ldots$ Then for any $T$, Bayes guarantees*

$$\sum_{m=1}^{M} U(m) \left( \sum_{t \leq T \,:\, m \in W_t} -\ln P(y_t|y_{<t}) - \sum_{t \leq T \,:\, m \in W_t} -\ln P(y_t|y_{<t}, m) \right) \leq \triangle \big( U(m) \big\| P(m) \big).$$

## 3 Sparse partition learning

We design efficient predictors with small regret compared to the best sparse partition model. We do this by constructing partition specialists from the input models and obtain a proper Bayesian predictor by averaging their predictions. We consider two priors. With the first prior we obtain the Mixing Past Posteriors (MPP) algorithm, giving it a Bayesian interpretation and slightly improving its regret bound. We then develop a new Markov chain prior. Bayes with this prior collapses to an efficient algorithm for which we prove the best known regret bound compared to sparse partitions.

**Construction** Each *partition specialist* $(\chi, m)$ is parameterized by a model index $m$ and a *circadian* (wake/sleep pattern) $\chi = (\chi_1, \chi_2, \ldots)$ with $\chi_t \in \{\mathsf{w}, \mathsf{s}\}$. We use infinite circadians in order to obtain algorithms that do not depend on a time horizon. The wake set $W_t$ includes all partition specialists that are awake at time $t$, i.e. $W_t := \{(\chi, m) \mid \chi_t = \mathsf{w}\}$. An awake specialist $(\chi, m)$ in $W_t$ predicts as the base model $m$, i.e. $\mathcal{P}(y_t|y_{<t}, (\chi, m)) := P(y_t|y_{<t}, m)$. The Bayesian joint distribution $\mathcal{P}$ is completed[2] by choosing a prior on partition specialists. In this paper we enforce the independence $\mathcal{P}(\chi, m) := \mathcal{P}(\chi)\mathcal{P}(m)$ and define $\mathcal{P}(m) := 1/M$ uniform on the base models. We now can apply Theorem 1 to bound the regret w.r.t. any partition model with time horizon $T$ by decomposing it into $N$ partition specialists $(\chi^1_{\leq T}, \hat{m}^1), \ldots, (\chi^N_{\leq T}, \hat{m}^N)$ and choosing $U(\cdot) = 1/N$ uniform on these specialists:

$$\mathcal{R} \leq N \ln \frac{M}{N} + \sum_{n=1}^{N} -\ln \mathcal{P}(\chi^n_{\leq T}). \quad (3)$$

The overhead of selecting $N$ reference models from the pool of size $M$ closely approximates the information-theoretic ideal $N \ln \frac{M}{N} \approx \ln \binom{M}{N}$. This improves previous regret bounds [BW02, ABR07, CBGLS12] by an additive $N \ln N$. Next we consider two choices for $\mathcal{P}(\chi)$: one for which we retrieve MPP, and a natural one which leads to efficient algorithms and sharper bounds.

### 3.1 A circadian prior equivalent to Mixing Past Posteriors

The Mixing Past Posteriors algorithm is parameterized a so-called *mixing scheme*, which is a sequence $\gamma_1, \gamma_2, \ldots$ of distributions, each $\gamma_t$ with support $\{0, \ldots, t-1\}$. MPP predicts outcome $y_t$ with $\mathrm{Pred}_t(y_t) := \sum_{m=1}^{M} P(y_t|y_{<t}, m) \, v_t(m)$, i.e. by averaging the model predictions with weights $v_t(m)$ defined recursively by

$$v_t(m) := \sum_{s=0}^{t-1} \tilde{v}_{s+1}(m) \, \gamma_t(s) \quad \text{where} \quad \tilde{v}_1(m) := \frac{1}{M} \quad \text{and} \quad \tilde{v}_{t+1}(m) := \frac{P(y_t|y_{<t}, m) v_t(m)}{\mathrm{Pred}_t(y_t)}.$$

The auxiliary distribution $\tilde{v}_{t+1}(m)$ is formally the (incremental) posterior from prior $v_t(m)$. The predictive weights $v_t(m)$ are then the pre-specified $\gamma_t$ mixture of all such past posteriors.

To make the Bayesian predictor equal to MPP, we define from the MPP mixing scheme a circadian prior measure $\mathcal{P}(\chi)$ that puts mass only on sequences with a finite nonzero number of w's, by

$$\mathcal{P}(\chi) := \frac{1}{s_J(s_J+1)} \prod_{j=1}^{J} \gamma_{s_j}(s_{j-1}) \quad \text{where } s_{\leq J} \text{ are the indices of the w's in } \chi \text{ and } s_0 = 0. \quad (4)$$

We built the independence $m \perp \chi$ into the prior $\mathcal{P}(\chi, m)$ and (4) ensures $\chi_{<t} \perp \chi_{>t} \mid \chi_t = \mathsf{w}$ for all $t$. Since the outcomes $y_{\leq t}$ are a stochastic function of $m$ and $\chi_{\leq t}$, the Bayesian joint satisfies

$$y_{\leq t}, m \perp \chi_{>t} \mid \chi_t = \mathsf{w} \qquad \text{for all } t. \quad (5)$$

**Theorem 2.** *Let* $\mathrm{Pred}_t(y_t)$ *be the prediction of MPP for some mixing scheme* $\gamma_1, \gamma_2, \ldots$ *Let* $\mathcal{P}(y_t|y_{<t})$ *be the prediction of Bayes with prior* (4). *Then for all outcomes* $y_{\leq t}$

$$\mathrm{Pred}_t(y_t) = \mathcal{P}(y_t|y_{<t}).$$

*Proof.* Partition the event $W_t = \{\chi_t = \mathsf{w}\}$ into $Z_q^t := \{\chi_t = \chi_q = \mathsf{w} \text{ and } \chi_r = \mathsf{s} \text{ for all } q < r < t\}$ for all $0 \leq q < t$, with the convention that $\chi_0 = \mathsf{w}$. We first establish that the Bayesian joint with prior (4) satisfies $y_{\leq t} \perp W_t$ for all $t$. Namely, by induction on $t$, for all $q < t$

$$\mathcal{P}(y_{<t}|Z_q^t) = \mathcal{P}(y_{<t}|y_{\leq q})\mathcal{P}(y_{\leq q}|Z_q^t) \overset{(5)}{=} \mathcal{P}(y_{<t}|y_{\leq q})\mathcal{P}(y_{\leq q}|W_q) \overset{\text{Induction}}{=} \mathcal{P}(y_{<t}),$$

and therefore $\mathcal{P}(y_{\leq t}|W_t) = \mathcal{P}(y_t|y_{<t}) \sum_{q=0}^{t-1} \mathcal{P}(y_{<t}|Z_q^t)\mathcal{P}(Z_q^t|W_t) = \mathcal{P}(y_{\leq t})$, i.e. $y_{\leq t} \perp W_t$. The theorem will be implied by the stronger claim $v_t(m) = \mathcal{P}(m|y_{<t}, W_t)$, which we again prove by induction on $t$. The case $t = 1$ is trivial. For $t > 1$, we expand the right-hand side, apply (5), use the independence we just proved, and the fact that asleep specialist predict with the rest:

$$\mathcal{P}(m|y_{<t}, W_t) = \sum_{q=0}^{t-1} \mathcal{P}(m|y_{\leq q}, W_q) \frac{\mathcal{P}(Z_q^t|\cancel{m, y_{\leq q}}, W_q)\mathcal{P}(W_q|y_{\leq q})}{\mathcal{P}(W_t|\cancel{y_{<t}})} \frac{\mathcal{P}(y_{<t}|Z_q^t, \cancel{m, y_{\leq q}})}{\cancel{\mathcal{P}(y_{<t}|y_{\leq q})}}$$

$$= \sum_{q=0}^{t-1} \frac{\mathcal{P}(y_q|y_{<q}, m)\mathcal{P}(m|y_{<q}, W_q)}{\mathcal{P}(y_q|y_{<q})} \mathcal{P}(Z_q^t|W_t)$$

By (4) $\mathcal{P}(Z_q^t|W_t) = \gamma_t(q)$, and the proof is completed by applying the induction hypothesis. □

The proof of the theorem provides a Bayesian interpretation of all the MPP weights: $v_t(m) = \mathcal{P}(m|y_{<t}, W_t)$ is the predictive distribution, $\tilde{v}_{t+1}(m) = \mathcal{P}(m|y_{\leq t}, W_t)$ is the posterior, and $\gamma_t(q) = \mathcal{P}(Z_q^t|W_t)$ is the conditional probability of the previous awake time.

## 3.2 A simple Markov chain circadian prior

In the previous section we recovered circadian priors corresponding to the MPP mixing schemes. Here we design priors afresh from first principles. Our goal is efficiency and good regret bounds. A simple and intuitive choice for prior $\mathcal{P}(\chi)$ is a Markov chain on states $\{\mathsf{w}, \mathsf{s}\}$ with initial distribution $\theta(\cdot)$ and transition probabilities $\theta(\cdot|\mathsf{w})$ and $\theta(\cdot|\mathsf{s})$, that is

$$\mathcal{P}(\chi_{\leq t}) := \theta(\chi_1) \prod_{s=2}^{t} \theta(\chi_s|\chi_{s-1}). \quad (6)$$

By choosing low transition probabilities we obtain a prior that favors temporal locality in that it allocates high probability to circadians that are awake and asleep in contiguous segments. Thus if a good sparse partition model exists for the data, our algorithm will pick up on this and predict well.

The resulting Bayesian strategy (aggregating infinitely many specialists) can be executed efficiently.

**Theorem 3.** *The prediction* $\mathcal{P}(y_t|y_{<t})$ *of Bayes with Markov prior* (6) *equals the prediction* $\mathrm{Pred}_t(y_t)$ *of Algorithm 1, which can be computed in* $\mathcal{O}(M)$ *time per outcome using* $\mathcal{O}(M)$ *space.*

*Proof.* We prove by induction on $t$ that $v_t(b,m) = \mathcal{P}(\chi_t = b, m|y_{<t})$ for each model $m$ and $b \in \{\mathsf{w}, \mathsf{s}\}$. The base case $t = 1$ is automatic. For the induction step we expand

$$\mathcal{P}(\chi_{t+1} = b, m|y_{\leq t}) \overset{(6)}{=} \theta(b|\mathsf{w})\mathcal{P}(\chi_t = \mathsf{w}, m|y_{\leq t}) + \theta(b|\mathsf{s})\mathcal{P}(\chi_t = \mathsf{s}, m|y_{\leq t})$$

$$\overset{(2)}{=} \theta(b|\mathsf{w}) \frac{\mathcal{P}(\chi_t = \mathsf{w}, m|y_{<t})P(y_t|y_{<t}, m)}{\sum_{i=1}^{M} \mathcal{P}(i|\chi_t = \mathsf{w}, y_{<t})P(y_t|y_{<t}, i)} + \theta(b|\mathsf{s})\mathcal{P}(\chi_t = \mathsf{s}, m|y_{<t}).$$

By applying the induction hypothesis we obtain the update rule for $v_{t+1}(b,m)$. $\qquad\square$

---

**Algorithm 1** Bayes with Markov circadian prior (6) (for Freund's problem)

---

**Input:** Distributions $\theta(\cdot)$, $\theta(\cdot|\mathsf{w})$ and $\theta(\cdot|\mathsf{s})$ on $\{\mathsf{w}, \mathsf{s}\}$.
Initialize $v_1(b,m) := \theta(b)/M$ for each model $m$ and $b \in \{\mathsf{w}, \mathsf{s}\}$
**for** $t = 1, 2, \ldots$ **do**
 Receive prediction $P(y_t|y_{<t}, m)$ of each model $m$
 Predict with $\mathrm{Pred}_t(y_t) := \sum_{m=1}^{M} P(y_t|y_{<t}, m)v_t(m|\mathsf{w})$ where $v_t(m|\mathsf{w}) = \frac{v_t(\mathsf{w}, m)}{\sum_{m'=1}^{M} v_t(\mathsf{w}, m')}$
 Observe outcome $y_t$ and suffer loss $-\ln \mathrm{Pred}_t(y_t)$.
 Update $v_{t+1}(b, m) := \theta(b|\mathsf{w})\frac{P(y_t|y_{<t}, m)}{\mathrm{Pred}_t(y_t)} v_t(\mathsf{w}, m) + \theta(b|\mathsf{s})v_t(\mathsf{s}, m)$.
**end for**

---

The previous theorem establishes that we can predict fast. Next we show that we predict well.

**Theorem 4.** *Let $\hat{m}_1, \ldots, \hat{m}_T$ be an $N$-sparse assignment of $M$ models to $T$ times with $B$ segments. The regret of Bayes (Algorithm 1) with tuning $\theta(\mathsf{w}) = 1/N$, $\theta(\mathsf{s}|\mathsf{w}) = \frac{B-1}{T-1}$ and $\theta(\mathsf{w}|\mathsf{s}) = \frac{B-1}{(N-1)(T-1)}$ is at most*

$$\mathcal{R} \leq N \ln \frac{M}{N} + N\mathcal{H}\left(\frac{1}{N}\right) + (T-1)\mathcal{H}\left(\frac{B-1}{T-1}\right) + (N-1)(T-1)\mathcal{H}\left(\frac{B-1}{(N-1)(T-1)}\right),$$

*where $\mathcal{H}(p) := -p\ln(p) - (1-p)\ln(1-p)$ is the binary entropy function.*

*Proof.* Without generality assume $\hat{m}_t \in \{1, \ldots, N\}$. For each reference model $n$ pick circadian $\chi_{\leq T}^{n}$ with $\chi_t^{n} = \mathsf{w}$ iff $\hat{m}_t = n$. Expanding the definition of the prior (6) we find

$$\prod_{n=1}^{N} \mathcal{P}(\chi_{\leq T}^{n}) = \theta(\mathsf{w})\theta(\mathsf{s})^{N-1}\theta(\mathsf{s}|\mathsf{s})^{(N-1)(T-1)-(B-1)}\theta(\mathsf{w}|\mathsf{w})^{T-B}\theta(\mathsf{w}|\mathsf{s})^{B-1}\theta(\mathsf{s}|\mathsf{w})^{B-1},$$

which is in fact maximized by the proposed tuning. The theorem follows from (3). $\qquad\square$

---

The information-theoretic ideal regret is $\ln\binom{M}{N} + \ln\binom{T-1}{B-1} + B\ln N$. Theorem 4 is very close to this except for a factor of 2 in front of the middle term; since $n\mathcal{H}(k/n) \leq k\ln(n/k) + k$ we have

$$\mathcal{R} \leq N\ln\frac{M}{N} + \textcircled{2}\ (B-1)\ln\frac{T-1}{B-1} + B\ln N + 2B.$$

The origin of this factor remained a mystery in [BW02], but becomes clear in our analysis: it is the *price of coordination* between the specialists that constitute the best partition model. To see this, let us regard a circadian as a sequence of wake/sleep transition times. With this viewpoint (3) bounds the regret by summing the prior costs of all the reference wake/sleep transition times. This means that we incur overhead at each segment boundary of the comparator *twice*: once as the sleep time of the preceding model, and once more as the wake time of the subsequent model.

In practice the comparator parameters $T$, $N$ and $B$ are unknown. This can be addressed by standard orthogonal techniques. Of particular interest is the method inspired by [SM99, KdR08, Koo11] of changing the Markov transition probabilities as a function of time. It can be shown that by setting $\theta(\mathsf{w}) = 1/2$ and increasing $\theta(\mathsf{w}|\mathsf{w})$ and $\theta(\mathsf{s}|\mathsf{s})$ as $\exp(-\frac{1}{t\ln^2(t+1)})$ we keep the update time and space of the algorithm at $\mathcal{O}(M)$ and guarantee regret bounded for all $T$, $N$ and $B$ as

$$\mathcal{R} \leq N\ln\frac{M}{N} + 2N + 2(B-1)\ln T + 4(B-1)\ln\ln(T+1).$$

At no computational overhead, this bound is remarkably close to the fully tuned bound of the theorem above, especially when the number of segments $B$ is modest as a function of $T$.

# 4  Sparse multitask learning

We transition to an extension of the sequential prediction setup called online multitask learning [ABR07, RAB07, ARB08, LPS09, CCBG10, SRDV11]. The new ingredient is that before predicting outcome $y_t$ we are given its *task number* $\kappa_t \in \{1, \ldots, K\}$. The goal is to exploit similarities between tasks. As before, we have access to $M$ models that each issue a prediction each round. If a single model predicts well on several tasks we want to figure this out quickly and exploit it. Simply ignoring the task number would not result in an adaptive algorithm. Applying a separate Bayesian predictor to each task independently would not result in any inter-task synergy. Nevertheless, it would guarantee regret at most $K \ln M$ overall. Now suppose each task is predicted well by some model from a small subset of models of size $N \ll M$. Running Bayes on all $N$-sparse allocations would achieve regret $\ln \binom{M}{N} + K \ln N$. However, emulating Bayes in this case is NP-hard [RAB07]. The goal is to design efficient algorithms with approximately the same regret bound.

In [ABR07] this multiclass problem is reduced to MPP, giving regret bound $N \ln \frac{M}{N} + B \ln N$. Here $B$ is the number of same-task segments in the task sequence $\kappa_{\leq T}$. When all outcomes with the same task number are consecutive, i.e. $B = K$, then the desired bound is achieved. However the tasks may be interleaved, making the number of segments $B$ much larger than $K$. We now eliminate the dependence on $B$, i.e. we solve a key open problem of [ABR07].

We apply the method of specialists to multitask learning, and obtain regret bounds close to the information-theoretic ideal, which in particular do not depend on the task segment count $B$ at all.

**Construction**   We create a *subset specialist* $(S, m)$ for each basic model index $m$ and subset of tasks $S \subseteq \{1, \ldots, K\}$. At time $t$, specialists with the current task $\kappa_t$ in their set $S$ are awake, i.e. $W_t := \{(S, m) \mid \kappa_t \in S\}$, and issue the prediction $\mathcal{P}(y_t | y_{<t}, S, m) := P(y_t | y_{<t}, m)$ of model $m$. We assign to subset specialist $(S, m)$ prior probability $\mathcal{P}(S, m) := \mathcal{P}(S)\mathcal{P}(m)$ where $\mathcal{P}(m) := 1/M$ is uniform, and $\mathcal{P}(S)$ includes each task independently with some fixed bias $\sigma(\mathsf{w})$

$$\mathcal{P}(S) := \sigma(\mathsf{w})^{|S|} \sigma(\mathsf{s})^{K-|S|}. \tag{7}$$

This construction has the property that the product of prior weights of two loners $(\{\kappa_1\}, \hat{m})$ and $(\{\kappa_2\}, \hat{m})$ is dramatically lower than the single pair specialist $(\{\kappa_1, \kappa_2\}, \hat{m})$, especially so when the number of models $M$ is large or when we consider larger task clusters. By strongly favoring it in the prior, any inter-task similarity present will be picked up fast.

The resulting Bayesian strategy involving $M2^K$ subset specialists can be implemented efficiently.

**Theorem 5.** *The predictions $\mathcal{P}(y_t | y_{<t})$ of Bayes with the set prior* (7) *equal the predictions* $\mathrm{Pred}_t(y_t)$ *of Algorithm 2. They can be computed in $\mathcal{O}(M)$ time per outcome using $\mathcal{O}(KM)$ storage.*

Of particular interest is Algorithm 2's update rule for $f_{t+1}^\kappa(m)$. This would be a regular Bayesian posterior calculation if $v_t(m)$ in $\mathrm{Pred}_t(y_t)$ were replaced by $f_t^\kappa(m)$. In fact, $v_t(m)$ is the communication channel by which knowledge about the performance of model $m$ in other tasks is received.

*Proof.* The resource analysis follows from inspection, noting that the update is fast because only the weights $f_t^\kappa(m)$ associated to the current task $\kappa$ are changed. We prove by induction on $t$ that $\mathcal{P}(m | y_{<t}, W_t) = v_t(m)$. In the base case $t = 1$ both equal $1/M$. For the induction step we expand $\mathcal{P}(m | y_{\leq t}, W_{t+1})$, which is by definition proportional to

$$\sum_{S \ni \kappa_{t+1}} \frac{1}{M} \sigma(\mathsf{w})^{|S|} \sigma(\mathsf{s})^{K-|S|} \left( \prod_{q \leq t \,:\, \kappa_q \in S} P(y_q | y_{<q}, m) \right) \left( \prod_{q \leq t \,:\, \kappa_q \notin S} \mathcal{P}(y_q | y_{<q}) \right). \tag{8}$$

The product form of both set prior and likelihood allows us to factor this exponential sum of products into a product of binary sums. It follows from the induction hypothesis that

$$f_t^k(m) = \frac{\sigma(\mathsf{w})}{\sigma(\mathsf{s})} \prod_{q \leq t \,:\, \kappa_q = k} \frac{P(y_q | y_{<q}, m)}{\mathcal{P}(y_q | y_{<q})}$$

Then we can divide (8) by $\mathcal{P}(y_{\leq t}) \sigma(\mathsf{s})^K$ and reorganize to

$$\mathcal{P}(m | y_{\leq t}, W_{t+1}) \propto \frac{1}{M} f_t^{\kappa_{t+1}}(m) \prod_{k \neq \kappa_{t+1}} \left( f_t^k(m) + 1 \right) = \frac{1}{M} \frac{f_t^{\kappa_{t+1}}(m)}{f_t^{\kappa_{t+1}}(m) + 1} \prod_{k=1}^{K} \left( f_t^k(m) + 1 \right)$$

Since the algorithm maintains $\pi_t(m) = \prod_{k=1}^K (f_t^k(m) + 1)$ this is proportional to $v_{t+1}(m)$. $\qquad \square$

---

**Algorithm 2** Bayes with set prior (7) (for online multitask learning)

---

**Input:** Number of tasks $K \geq 2$, distribution $\sigma(\cdot)$ on $\{\mathsf{w}, \mathsf{s}\}$.
Initialize $f_1^k(m) := \frac{\sigma(\mathsf{w})}{\sigma(\mathsf{s})}$ for each task $k$ and $\pi_1(m) := \prod_{k=1}^K (f_1^k(m) + 1)$.
**for** $t = 1, 2, \ldots$ **do**
    Observe task index $\kappa = \kappa_t$.
    Compute auxiliary $v_t(m) := \frac{f_t^\kappa(m)\, \pi_t(m)/(f_t^\kappa(m)+1)}{\sum_{i=1}^M f_t^\kappa(i)\, \pi_t(i)/(f_t^\kappa(i)+1)}$.
    Receive prediction $P(y_t|y_{<t}, m)$ of each model $m$
    Issue prediction $\mathrm{Pred}_t(y_t) := \sum_{m=1}^M P(y_t|y_{<t}, m)v_t(m)$.
    Observe outcome $y_t$ and suffer loss $-\ln \mathrm{Pred}_t(y_t)$.
    Update $f_{t+1}^\kappa(m) := \frac{P(y_t|y_{<t}, m)}{\mathrm{Pred}_t(y_t)} f_t^\kappa(m)$ and keep $f_{t+1}^k(m) := f_t^k(m)$ for all $k \neq \kappa$.
    Update $\pi_{t+1}(m) := \frac{f_{t+1}^\kappa(m)+1}{f_t^\kappa(m)+1} \pi_t(m)$.
**end for**

---

The Bayesian strategy is hence emulated fast by Algorithm 2. We now show it predicts well.

**Theorem 6.** *Let $\hat{m}_1, \ldots, \hat{m}_K$ be an $N$-sparse allocation of $M$ models to $K$ tasks. With tuned inclusion rate $\sigma(\mathsf{w}) = 1/N$, the regret of Bayes (Algorithm 2) is bounded by*

$$\mathcal{R} \;\leq\; N \ln\left(M/N\right) + KN\, \mathcal{H}(1/N).$$

*Proof.* Without loss of generality assume that $\hat{m}_k \in \{1, \ldots, N\}$. Let $S_n := \{1 \leq k \leq K \mid \hat{m}_k = n\}$. The sets $S_n$ for $n = 1, \ldots, N$ form a partition of the $K$ tasks. By (7) $\prod_{n=1}^N \mathcal{P}(S_n) = \sigma(\mathsf{w})^K \sigma(\mathsf{s})^{(N-1)K}$, which is maximized by the proposed tuning. The theorem now follows from (3). $\qquad \square$

We achieve the desired goal since $KN\, \mathcal{H}(1/N) \approx K \ln N$. In practice $N$ is of course unavailable for tuning, and we may tune $\sigma(\mathsf{w}) = 1/K$ pessimistically to get $K \ln K + N$ instead for all $N$ simultaneously. Or alternatively, we may sacrifice some time efficiency to externally mix over all $M$ possible values with decreasing prior, increasing the tuned regret by just $\ln N + \mathcal{O}(\ln \ln N)$. If in addition the number of tasks is unknown or unbounded, we may (as done in Section 3.2) decrease the membership rate $\sigma(\mathsf{w})$ with each new task encountered and guarantee regret $\mathcal{R} \leq N \ln(M/N) + K \ln K + 4N + 2K \ln \ln K$ where now $K$ is the number of tasks actually received.

## 5 Discussion

We showed that *Mixing Past Posteriors* is not just a heuristic with an unusual regret bound: we gave it a full Bayesian interpretation using specialist models. We then applied our method to a multitask problem. Again an unusual algorithm resulted that exploits sparsity by pulling up the weights of models that have done well before in other tasks. In other words, if all tasks are well predicted by a small subset of base models, then this algorithm improves its prior over models as it learns from previous tasks. Both algorithms closely circumvent NP-hardness. The deep question is whether some of the common updates used in Nature can be brought into the Bayesian fold using the specialist mechanism.

There are a large number of more immediate technical open problems (we just discuss a few). We presented our results using probabilities and log loss. However the bounds should easily carry over to the typical pseudo-likelihoods employed in online learning in connection with other loss functions. Next, it would be worthwhile to investigate for which infinite sets of models we can still employ our updates implicitly. It was already shown in [KvE10, Koo11] that MPP can be efficiently emulated on all Bernoulli models. However, what about Gaussians, exponential families in general, or even linear regression? Finally, is there a Bayesian method for modeling concurrent multitasking, i.e. can the Bayesian analysis be generalized to the case where a small subset of models solve many tasks *in parallel*?

## Footnotes

[1]The experiments reported in [HLSS00] are based on precursors of MPP. However MPP outperforms these algorithms in later experiments we have done on natural data for the same problem (not shown).

[2]From here on we use the symbol $\mathcal{P}$ for the Bayesian joint to avoid a fundamental ambiguity: $\mathcal{P}(y_t|y_{<t}, m)$ does *not* equal the prediction $P(y_t|y_{<t}, m)$ of the input model $m$, since it averages over both asleep and awake specialists $(\chi, m)$. The predictions of base models are now recovered as $\mathcal{P}(y_t|y_{<t}, W_t, m) = P(y_t|y_{<t}, m)$.

# References

[ABR07]    Jacob Ducan Abernethy, Peter Bartlett, and Alexander Rakhlin. Multitask learning with expert advice. Technical report, University of California at Berkeley, January 2007.

[ARB08]    Alekh Agarwal, Alexander Rakhlin, and Peter Bartlett. Matrix regularization techniques for online multitask learning, October 2008.

[BW02]    Olivier Bousquet and Manfred K. Warmuth. Tracking a small set of experts by mixing past posteriors. *Journal of Machine Learning Research*, 3:363–396, 2002.

[CBGLS12] Nicolò Cesa-Bianchi, Pierre Gaillard, Gábor Lugosi, and Gilles Stoltz. A new look at shifting regret. *CoRR*, abs/1202.3323, 2012.

[CCBG10]    Giovanni Cavallanti, Nicolò Cesa-Bianchi, and Claudio Gentile. Linear algorithms for online multitask classification. *J. Mach. Learn. Res.*, 11:2901–2934, December 2010.

[CKZV10]    Alexey Chernov, Yuri Kalnishkan, Fedor Zhdanov, and Vladimir Vovk. Supermartingales in prediction with expert advice. *Theor. Comput. Sci.*, 411(29-30):2647–2669, June 2010.

[CV09]    Alexey Chernov and Vladimir Vovk. Prediction with expert evaluators' advice. In *Proceedings of the 20th international conference on Algorithmic learning theory*, ALT'09, pages 8–22, Berlin, Heidelberg, 2009. Springer-Verlag.

[FSSW97]    Y. Freund, R. E. Schapire, Y. Singer, and M. K. Warmuth. Using and combining predictors that specialize. In *Proc. 29th Annual ACM Symposium on Theory of Computing*, pages 334–343. ACM, 1997.

[GWBA02]    Robert B. Gramacy, Manfred K. Warmuth, Scott A. Brandt, and Ismail Ari. Adaptive caching by refetching. In Suzanna Becker, Sebastian Thrun, and Klaus Obermayer, editors, *NIPS*, pages 1465–1472. MIT Press, 2002.

[HLSS00]    David P. Helmbold, Darrell D. E. Long, Tracey L. Sconyers, and Bruce Sherrod. Adaptive disk spin-down for mobile computers. *ACM/Baltzer Mobile Networks and Applications (MONET)*, pages 285–297, 2000.

[HW98]    Mark Herbster and Manfred K. Warmuth. Tracking the best expert. *Machine Learning*, 32:151–178, 1998.

[KdR08]    Wouter M. Koolen and Steven de Rooij. Combining expert advice efficiently. In Rocco Servedio and Tong Zang, editors, *Proceedings of the 21st Annual Conference on Learning Theory (COLT 2008)*, pages 275–286, June 2008.

[Koo11]    Wouter M. Koolen. *Combining Strategies Efficiently: High-quality Decisions from Conflicting Advice*. PhD thesis, Institute of Logic, Language and Computation (ILLC), University of Amsterdam, January 2011.

[KvE10]    Wouter M. Koolen and Tim van Erven. Freezing and sleeping: Tracking experts that learn by evolving past posteriors. *CoRR*, abs/1008.4654, 2010.

[LPS09]    Gábor Lugosi, Omiros Papaspiliopoulos, and Gilles Stoltz. Online multi-task learning with hard constraints. In *COLT*, 2009.

[RAB07]    Alexander Rakhlin, Jacob Abernethy, and Peter L. Bartlett. Online discovery of similarity mappings. In *Proceedings of the 24th international conference on Machine learning*, ICML '07, pages 767–774, New York, NY, USA, 2007. ACM.

[SM99]    Gil I. Shamir and Neri Merhav. Low complexity sequential lossless coding for piecewise stationary memoryless sources. *IEEE Trans. Info. Theory*, 45:1498–1519, 1999.

[SRDV11]    Avishek Saha, Piyush Rai, Hal Daumé III, and Suresh Venkatasubramanian. Online learning of multiple tasks and their relationships. In *AISTATS*, Ft. Lauderdale, Florida, 2011.

[VW98]    Paul A.J. Volf and Frans M.J. Willems. Switching between two universal source coding algorithms. In *Proceedings of the Data Compression Conference, Snowbird, Utah*, pages 491–500, 1998.

